# Merging Constrained Optimisation with Deterministic Annealing to "Solve" Combinatorially Hard Problems

**Paul Stolorz***

Santa Fe Institute
1660 Old Pecos Trail, Suite A
Santa Fe, NM 87501

## ABSTRACT

Several parallel analogue algorithms, based upon mean field theory (MFT) approximations to an underlying statistical mechanics formulation, and requiring an externally prescribed annealing schedule, now exist for finding approximate solutions to difficult combinatorial optimisation problems. They have been applied to the Travelling Salesman Problem (TSP), as well as to various issues in computational vision and cluster analysis. I show here that any given MFT algorithm can be combined in a natural way with notions from the areas of constrained optimisation and adaptive simulated annealing to yield a single homogenous and efficient parallel relaxation technique, for which an externally prescribed annealing schedule is no longer required. The results of numerical simulations on 50-city and 100-city TSP problems are presented, which show that the ensuing algorithms are typically an order of magnitude faster than the MFT algorithms alone, and which also show, on occasion, superior solutions as well.

## 1  INTRODUCTION

Several promising parallel analogue algorithms, which can be loosely described by the term "deterministic annealing", or "mean field theory (MFT) annealing", have

recently been proposed as heuristics for tackling difficult combinatorial optimisation problems [1, 2, 3, 4, 5, 6, 7]. However, the annealing schedules must be imposed externally in a somewhat *ad hoc* manner in these procedures (although they can be made adaptive to a limited degree [8]). As a result, a number of authors [9, 10, 11] have considered the alternative analogue approach of Lagrangian relaxation, a form of constrained optimisation due originally to Arrow [12], as a different means of tackling these problems. The various alternatives require the introduction of a new set of variables, the Lagrange multipliers. Unfortunately, these usually lead in turn to either the inclusion of expensive penalty terms, or the consideration of restricted classes of problem constraints. The penalty terms also tend to introduce unwanted local minima in the objective function, and they must be included even when the algorithms are exact [13, 10]. These drawbacks prevent their easy application to large-scale combinatorial problems, containing 100 or more variables.

In this paper I show that the technical features of analogue mean field approximations can be merged with both Lagrangian relaxation methods, and with the broad philosophy of adaptive annealing without, importantly, requiring the large computational resources that typically accompany the Lagrangian methods. The result is a systematic procedure for crafting from any given MFT algorithm a single parallel homogeneous relaxation technique which needs no externally prescribed annealing schedule. In this way the computational power of the analogue heuristics is greatly enhanced. In particular, the Lagrangian framework can be used to construct an efficient adaptation of the elastic net algorithm [2], which is perhaps the most promising of the analogue heuristics. The results of numerical experiments are presented which display both increased computational efficiency, and on occasion, better solutions (avoidance of some local minima) over deterministic annealing. Also, the qualitative mechanism at the root of this behaviour is described. Finally, I note that the apparatus can be generalised to a procedure that uses several multipliers, in a manner that roughly parallels the notion of different temperatures at different physical locations in the simulated annealing heuristic.

## 2   DETERMINISTIC ANNEALING

The deterministic annealing procedures consist of tracking the local minimum of an objective function of the form

$$F(\underline{x}) = U(\underline{x}) - TS(\underline{x}) \tag{1}$$

where $\underline{x}$ represents the analogue variables used to describe the particular problem at hand, and $T \geq 0$ (initially chosen large) is an adjustable annealing, or temperature, parameter. As $T$ is lowered, the objective function undergoes a qualitative change from a convex to a distinctly non-convex function. Provided the annealing shedule is slow enough, however, it is hoped that the local minimum near $T = 0$ is a close approximation to the global solution of the problem.

The function $S(\underline{x})$ represents an analogue approximation [5, 4, 7] to the entropy of an underlying discrete statistical physics system, while $F(\underline{x})$ approximates its free energy. The underlying discrete system forms the basis of the simulated annealing heuristic [14]. Although a general and powerful technique, this heuristic is an inherently stochastic procedure which must consider many individual discrete tours at

each and every temperature $T$. The deterministic annealing approximations have the advantage of being deterministic, so that an approximate solution at a given temperature can be found with much less computational effort. In both cases, however, the complexity of the problem under consideration shows up in the need to determine with great care an annealing schedule for lowering the temperature parameter.

The primary contribution of this paper consists in pursuing the relationship between deterministic annealing and statistical physics one step further, by making explicit use of the fact that due to the statistical physics embedding of the deterministic annealing procedures,

$$S(\underline{x}_{min}) \to 0 \text{ as } T \to 0 \tag{2}$$

where $\underline{x}_{min}$ is the local minimum obtained for the parameter value $T$. This deceptively simple observation allows the consideration of the somewhat different approach of Lagrange multiplier methods to automatically determine a dynamics for $T$ in the analogue heuristics, using as a constraint the vanishing of the entropy function at zero temperature. This particular fact has not been explicitly used in any previous optimisation procedures based upon Lagrange multipliers, although it is implicit in the work of [9]. Most authors have focussed instead on the syntactic constraints contained in the function $U(\underline{x})$ when incorporating Lagrange multipliers. As a result the issue of eliminating an external annealing schedule has not been directly confronted.

## 3   LAGRANGE MULTIPLIERS

Multiplier methods seek the critical points of a "Lagrangian" function

$$F(\underline{x}, \lambda) = U(\underline{x}) - \lambda S(\underline{x}) \tag{3}$$

where the notation of (1) has been retained, in accordance with the philosophy discussed above. The only difference is that the parameter $T$ has been replaced by a variable $\lambda$ (the Lagrange multiplier), which is to be treated on the same basis as the variables $\underline{x}$. By definition, the critical points of $F(\underline{x}, \lambda)$ obey the so-called Kuhn-Tucker conditions

$$\begin{aligned}
\nabla_{\underline{x}} F(\underline{x}, \lambda) &= 0 = \nabla_{\underline{x}} U(\underline{x}) - \lambda \nabla_{\underline{x}} S(\underline{x}) \\
\nabla_{\lambda} F(\underline{x}, \lambda) &= 0 = -S(\underline{x})
\end{aligned} \tag{4}$$

Thus, at any critical point of this function, the constraint $S(\underline{x}) = 0$ is satisfied. This corresponds to a vanishing entropy estimate in (1). Hopefully, in addition, $U(\underline{x})$ is minimised, subject to the constraint.

The difficulty with this approach when used in isolation is that finding the critical points of $F(\underline{x}, \lambda)$ entails, in general, the minimisation of a transformed "unconstrained" function, whose set of local minima contains the critical points of $F$ as a subset. This transformed function is required in order to ensure an algorithm which is convergent, because the critical points of $F(\underline{x}, \lambda)$ are saddle points, not local minima. One well-known way to do this is to add a term $S^2(\underline{x})$ to (3), giving an augmented Lagrangian with the same fixed points as (3), but hopefully with better convergence properties. Unfortunately, the transformed function is invariably more complicated than $F(\underline{x}, \lambda)$, typically containing extra quadratic penalty

terms (as in the above case), which tend to convert harmless saddle points into unwanted local minima. It also leads to greater computational overhead, usually in the form of either second derivatives of the functions $U(\underline{x})$ and $S(\underline{x})$, or of matrix inversions [13, 10] (although see [11] for an approach which minimises this overhead). For large-scale combinatorial problems such as the TSP these disadvantages become prohibitive. In addition, the entropic constraint functions occurring in deterministic annealing tend to be quite complicated nonlinear functions of the variables involved, often with peculiar behaviour near the constraint condition. In these cases (the Hopfield /Tank method is an example) a term quadratic in the entropy cannot simply be added to (3) in a straightforward way to produce a suitable augmented Lagrangian (of course, such a procedure *is* possible with several of the terms in the internal energy $U(\underline{x})$).

## 4   COMBINING BOTH METHODS

The best features of each of the two approaches outlined above may be retained by using the following modification of the original first-order Arrow technique:

$$
\begin{aligned}
\dot{x}_i &= -\nabla_{x_i}\hat{F}(\underline{x},\lambda) &&= -\nabla_{x_i}U(\underline{x}) + \lambda\nabla_{x_i}S(\underline{x}) &&(5) \\
\dot{\lambda} &= +\nabla_\lambda\hat{F}(\underline{x},\lambda) &&= -S(\underline{x}) + c/\lambda
\end{aligned}
$$

where $\hat{F}(\underline{x},\lambda)$ is a slightly modified "free energy" function given by

$$
\hat{F}(\underline{x},\lambda) = U(\underline{x}) - \lambda S(\underline{x}) + c\ln\lambda \tag{6}
$$

In these expressions, $c > 0$ is a constant, chosen small on the scale of the other parameters, and characterises the sole, inexpensive, penalty requirement. It is needed purely in order to ensure that $\lambda$ remain positive. In fact, in the numerical experiment that I will present, this penalty term for $\lambda$ was not even used - the algorithm was simply terminated at a suitably small value of $\lambda$.

The reason for insisting upon $\lambda > 0$, in contrast to most first-order relaxation methods, is that it ensures that the free energy objective function is bounded below with respect to the $\underline{x}$ variables. This in turn allows (5) to be proven locally convergent [15] using techniques discussed in [13]. Furthermore, the methods described by (5) are found empirically to be globally convergent as well. This feature is in fact the key to their computational efficiency, as it means that they need not be grafted onto more sophisticated and inefficient methods in order to ensure convergence. This behaviour can be traced to the fact that the "free energy" functions, while non-convex overall with respect to $\underline{x}$, are nevertheless convex over large volumes of the solution space. The point can be illustrated by the construction of an energy function similar to that used by Platt and Barr [9], which also displays the mechanism by which some of the unwanted local minima in deterministic annealing may be avoided. These issues are discussed further in Section 6.

The algorithms described above have several features which distinguish them from previous work. Firstly, the entropy estimate $S(\underline{x})$ has been chosen explicitly as the appropriate constraint function, a fact which has previously been unexploited in the optimisation context (although a related piecewise linear function has been used by [9]). Further, since this estimate is usually positive for the mean field theory

heuristics, $\lambda$ (the only new variable) decreases monotonically in a manner roughly similar to the temperature decrease schedule used in simulated and deterministic annealing, but with the *ad hoc* drawback now removed. Moreover, there is no requirement that the system be at or near a fixed point each time $\lambda$ is altered - there is simply one homogeneous dynamical system which must approach a fixed point only once at the very end of the simulation, and furthermore $\lambda$ appears linearly except near the end of the procedure (a major reason for its efficiency). Finally, the algorithms do not require computationally cumbersome extra structure in the form of quadratic penalty terms, second derivatives or inverses, in contrast to the usual Lagrangian relaxation techniques. All of these features can be seen to be due to the statistical physics setting of the annealing "Lagrangian", and the use of an entropic constraint instead of the more usual syntactic constraints.

The apparatus outlined above can immediately be used to adapt the Hopfield/Tank heuristic for the Travelling Salesman Problem (TSP) [1], which can easily be written in the form (1). However, the elastic net method [2] is known to be a somewhat superior method, and is therefore a better candidate for modification. There is an impediment to the procedure here: the objective function for the elastic net is actually of the form

$$F(\underline{x}, \lambda) = U(\underline{x}) - \lambda S(\underline{x}, \lambda) \tag{7}$$

which precludes the use of a true Lagrange muliplier, since $\lambda$ now appears non-trivially in the constraint function itself! However, I find surprisingly that the algorithm obtained by applying the Lagrangian relaxation apparatus in a straight-forward way as before still leads to a coherent algorithm. The equations are

$$\dot{x}_i = -\nabla_{x_i} F(\underline{x}, \lambda) = -\nabla_{x_i} U(\underline{x}) + \lambda \nabla_{x_i} S(\underline{x}) \tag{8}$$
$$\dot{\lambda} = +\epsilon \nabla_\lambda F(\underline{x}, \lambda) = -\epsilon[S(\underline{x}, \lambda) + \lambda \nabla_\lambda S(\underline{x}, \lambda)]$$

The parameter $\epsilon > 0$ is chosen so that an explicit barrier term for $\lambda$ can be avoided. It is the only remaining externally prescribed part of the former annealing schedule, and is fixed just once at the begining of the algorithm.

It can be shown that the global convergence of (8) is highly plausible in general (and seems to always occur in practice), as in the simpler case described by (5). Secondly, and most importantly, it can be shown that the constraints that are obeyed at the new fixed points satisfy the syntax of the original discrete problem [15]. The procedure is not limited to the elastic net method for the TSP. The mean field approximations discussed in [3, 4, 5] all behave in a similar way, and can therefore be adapted successfully to Lagrangian relaxation methods. The form of the elastic net entropy function suggests a further natural generalisation of the procedure. A different "multiplier" $\lambda_a$ can be assigned to each city $a$, each variable being responsible for satisfying a different additive component of the entropy constraint. The idea has an obvious parallel to the notion in simulated annealing of lowering the temperature in different geographical regions at different rates in response to the behaviour of the system. The number of extra variables required is a modest computational investment, since there are typically many more tour points than city points for a given implementation.

## 5  RESULTS FOR THE TSP

Numerical simulations were performed on various TSP instances using the elastic net method, the Lagrangian adaptation with a single global Lagrange multiplier, and the modification discussed above involving one Lagrange multiplier for each city. The results are shown in Table 1. The tours for the Lagrangian relaxation methods are about 0.5% shorter than those for the elastic net, although these differences are not yet at a statistically significant level. The differences in the computational requirements are, however, much more dramatic. No attempt has been made to optimise any of the techniques by using sophisticated descent procedures, although the size of the update step has been chosen to seperately optimise each method.

Table 1: Performance of heuristics described in the text on a set of 40 randomly distibuted 50-city instances of the TSP in the unit square. CPU times quoted are for a SUN SPARC Station 1+. $\alpha$ and $\beta$ are the standard tuning parameters [4].

| METHOD | $\alpha$ | $\beta$ | TOUR LENGTH | CPU(SEC) |
|---|---|---|---|---|
| Elastic net | 0.2 | 2.5 | $5.95 \pm 0.10$ | $260 \pm 33$ |
| Global multiplier | 0.4 | 2.5 | $5.92 \pm 0.09$ | $49 \pm 5$ |
| Local multipliers | 0.4 | 2.5 | $5.92 \pm 0.08$ | $82 \pm 12$ |

I have also been able to obtain a superior solution to the 100-city problem analysed by Durbin and Willshaw [2], namely a solution of length 7.746 [15] (c.f. length 7.783 for the elastic net) in a fraction of the time taken by elastic net annealing. This represents an improvement of roughly 0.5%. Although still about 0.5% longer than the best tour found by simulated annealing, this result is quite encouraging, because it was obtained with far less CPU time than simulated annealing, and in substantially less time than the elastic net: improvements upon solutions within about 1% of optimality typically require a substantial increase in CPU investment.

## 6  HOW IT WORKS - VALLEY ASCENT

Inspection of the solutions obtained by the various methods indicates that the multiplier schemes can sometimes exchange enough "inertial" energy to overcome the energy barriers which trap the annealing methods, thus offering better solutions as well as much-improved computational efficiency. This point is illustrated in Figure 1(a), which displays the evolution of the following function during the algorithm for a typical set of parameters:

$$E = \frac{1}{2}\sum_i \dot{x_i}^2 + \frac{1}{2}\dot{\lambda}^2 \qquad (9)$$

The two terms can be thought of as different components of an overall kinetic energy $E$. During the procedure, energy can be exchanged between these two components, so the function $E(t)$ does not decrease monotonically with time. This allows the system to occasionally escape from local minima. Nevertheless, after a long enough

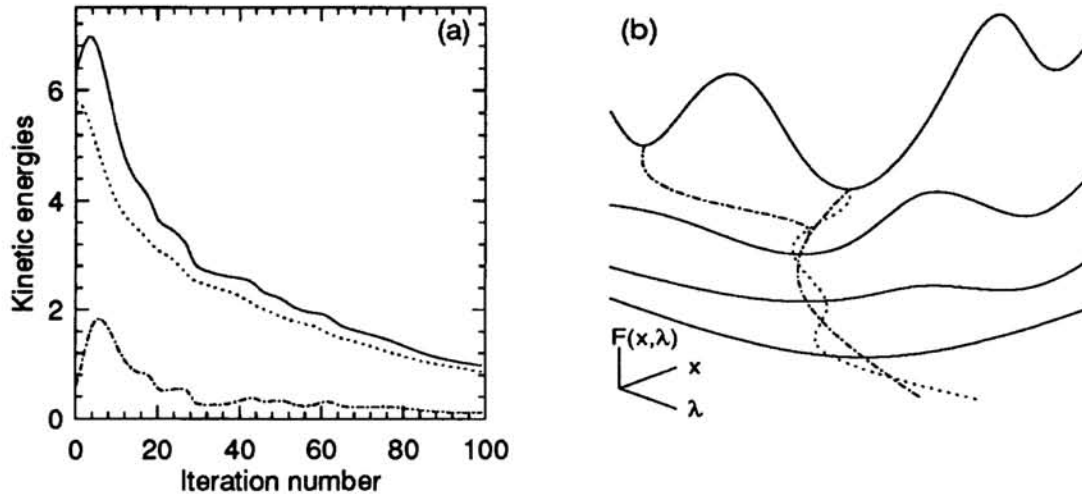

Figure 1: (a) Evolution of variables for a typical 50-city TSP. The solid curve shows the total kinetic energy $E$ given by (9). The dotted curve shows the $\lambda$ component of this energy, and the dash-dotted curve shows the $\underline{x}$ component. (b) Trajectories taken by various algorithms on a schematic free energy surface. The two dash-dotted curves show possible paths for elastic net annealing, each ascending a valley floor. The dotted curve shows a Lagrangian relaxation, which displays oscillations about the valley floor leading to the superior solution.

time the function does decrease smoothly, ensuring convergence to a valid solution to the problem.

The basic mechanism can also be understood by plotting schematically the free energy "surface" $F(\underline{x}, \lambda)$, as shown in Figure 1(b). This surface has a single valley in the foreground, where $\lambda$ is large. Bifurcations occur as $\lambda$ becomes smaller, with a series of saddles, each a valid problem solution, being reached in the background at $\lambda = 0$. Deterministic annealing can be viewed as the ascent of just one of these valleys along the valley floor. It is hoped that the broadest and deepest minimum is chosen at each valley bifurcation, leading eventually to the lowest background saddle point as the optimal solution. A typical trajectory for one of the Lagrangian modifications also consists roughly of the ascent of one of these valleys. However, oscillations about the valley floor now occur on the way to the final saddle point, due to the interplay between the different kinetic components displayed in Figure 1(a). It is hoped that the extra degrees of freedom allow valleys to be explored more fully near bifurcation points, thus biasing the larger valleys more than deterministic annealing. Notice that in order to generate the $\lambda$ dynamics, computational significance is now assigned to the actual *value* of the free energy in the new schemes, in contrast to the situation in regular annealing.

# 7    CONCLUSION

In summary, a simple yet effective framework has been developed for systematically generalising any algorithm described by a mean field theory approximation procedure to a Lagrangian method which replaces annealing by the relaxation of a single dynamical system. Even in the case of the elastic net, which has a slightly awkward

form, the resulting method can be shown to be sensible, and I find in fact that it substantially improves the speed (and accuracy) of that method. The adaptations depend crucially upon the vanishing of the analogue entropy at zero temperature. This allows the entropy to be used as a powerful constraint function, even though it is a highly nonlinear function and might be expected at first sight to be unsuitable for the task. In fact, this observation can also be applied in a wider context to design objective functions and architectures for neural networks which seek to improve generalisation ability by limiting the number of network parameters [16].

## Footnotes

*also at Theoretical Division and Center for Nonlinear Studies, MSB213, Los Alamos National Laboratory, Los Alamos, NM 87545.

# References

[1] J.J. Hopfield and D.W. Tank. Neural computation of decisions in optimization problems. *Biol. Cybern.*, 52:141–152, 1985.

[2] R. Durbin and D. Willshaw. An analogue approach to the travelling salesman problem using an elsatic net method. *Nature*, 326:689–691, 1987.

[3] D. Geiger and F. Girosi. Coupled markov random fields and mean field theory. In D. Touretzky, editor, *Advances in Neural Information Processing Systems 2*, pages 660–667. Morgan Kaufmann, 1990.

[4] A.L. Yuille. Generalised deformable models, statistical physics, and matching problems. *Neural Comp.*, 2:1–24, 1990.

[5] P.D. Simic. Statistical mechanics as the underlying theory of "elastic" and "neural" optimisations. *NETWORK: Comp. Neural Syst.*, 1:89–103, 1990.

[6] A. Blake and A. Zisserman. *Visual Reconstruction*. MIT Press, 1987.

[7] C. Peterson and B. Soderberg. A new method for mapping optimization problems onto neural networks. *Int. J. Neural Syst.*, 1:3–22, 1989.

[8] D.J. Burr. An improved elastic net method for the travelling salesman problem. In *IEEE 2nd International Conf. on Neural Networks*, pages I–69–76, 1988.

[9] J.C. Platt and A.H. Barr. Constrained differential optimization. In D.Z. Anderson, editor, *Neural Information Proc. Systems*, pages 612–621. AIP, 1988.

[10] A.G. Tsirukis, G.V. Reklaitis, and M.F. Tenorio. Nonlinear optimization using generalised hopfield networks. *Neural Comp.*, 1:511–521, 1989.

[11] E. Mjolsness and C. Garrett. Algebraic transformations of objective functions. *Neural Networks*, 3:651–669, 1990.

[12] K.J. Arrow, L. Hurwicz, and H. Uzawa. *Studies in Linear and Nonlinear Programming*. Stanford University Press, 1958.

[13] D.P. Bertsekas. *Constrained Optimization and Lagrange Multiplier Methods*. Academic Press, 1982. See especially Chapter 4.

[14] S. Kirkpatrick, C.D. Gelatt Jr., and M.P. Vecchi. Optimization by simulated annealing. *Science*, 220:671–680, 1983.

[15] P. Stolorz. Merging constrained optimisation with deterministic annealing to "solve" combinatorially hard problems. Technical report, LA-UR-91-3593, Los Alamos National Laboratory, 1991.

[16] P.Stolorz. Analogue entropy as a constraint in adaptive learning and optimisation. Technical report, in preparation, Santa Fe Institute, 1992.
